# Greedy importance sampling

**Dale Schuurmans**
Department of Computer Science
University of Waterloo
dale@cs.uwaterloo.ca

## Abstract

I present a simple variation of importance sampling that explicitly searches for important regions in the target distribution. I prove that the technique yields unbiased estimates, and show empirically it can reduce the variance of standard Monte Carlo estimators. This is achieved by concentrating samples in more significant regions of the sample space.

## 1 Introduction

It is well known that general inference and learning with graphical models is computationally hard [1] and it is therefore necessary to consider restricted architectures [13], or approximate algorithms to perform these tasks [3, 7]. Among the most convenient and successful techniques are stochastic methods which are guaranteed to converge to a correct solution in the limit of large samples [10, 11, 12, 15]. These methods can be easily applied to complex inference problems that overwhelm deterministic approaches.

The family of stochastic inference methods can be grouped into the *independent* Monte Carlo methods (importance sampling and rejection sampling [4, 10, 14]) and the *dependent* Markov Chain Monte Carlo (MCMC) methods (Gibbs sampling, Metropolis sampling, and "hybrid" Monte Carlo) [5, 10, 11, 15]. The goal of all these methods is to simulate drawing a random sample from a target distribution $P(x)$ (generally defined by a Bayesian network or graphical model) that is difficult to sample from directly.

This paper investigates a simple modification of importance sampling that demonstrates some advantages over independent and dependent-Markov-chain methods. The idea is to explicitly search for important regions in a target distribution $P$ when sampling from a simpler proposal distribution $Q$. Some MCMC methods, such as Metropolis and "hybrid" Monte Carlo, attempt to do something like this by biasing a local random search towards higher probability regions, while preserving the asymptotic "fair sampling" properties of the exploration [11, 12]. Here I investigate a simple direct approach where one draws points from a proposal distribution $Q$ but then explicitly searches in $P$ to find points from significant regions. The main challenge is to maintain correctness (*i.e.*, unbiasedness) of the resulting procedure, which we achieve by independently sampling search subsequences and then weighting the sample points so that their expected weight under the proposal distribution $Q$ matches their true probability under the target $P$.

**Importance sampling**
- Draw $x_1, ..., x_n$ independently from $Q$.
- Weight each point $x_i$ by $w(x_i) = \frac{P(x_i)}{Q(x_i)}$.
- For a random variable, $f$, estimate $\mathrm{E}_{P(x)} f(x)$
  by $\hat{f} = \frac{1}{n} \sum_{i=1}^{n} f(x_i) w(x_i)$.

**"Indirect" importance sampling**
- Draw $x_1, ..., x_n$ independently from $Q$.
- Weight each point $x_i$ by $u(x_i) = \frac{\beta P(x_i)}{Q(x_i)}$.
- For a random variable, $f$, estimate $\mathrm{E}_{P(x)} f(x)$
  by $\hat{f} = \sum_{i=1}^{n} f(x_i) u(x_i) / \sum_{i=1}^{n} u(x_i)$.

Figure 1: Regular and "indirect" importance sampling procedures

## 2 Generalized importance sampling

Many inference problems in graphical models can be cast as determining the expected value of a random variable of interest, $f$, given observations drawn according to a target distribution $P$. That is, we are interested in computing the expectation $\mathrm{E}_{P(x)} f(x)$. Usually the random variable $f$ is simple, like the indicator of some event, but the distribution $P$ is generally not in a form that we can sample from efficiently. *Importance sampling* is a useful technique for estimating $\mathrm{E}_{P(x)} f(x)$ in these cases. The idea is to draw independent points $x_1, ..., x_n$ from a simpler "proposal" distribution $Q$, but then weight these points by $w(x) = P(x)/Q(x)$ to obtain a "fair" representation of $P$. Assuming that we can efficiently evaluate $P(x)$ at each point, the weighted sample can be used to estimate desired expectations (Figure 1). The correctness (*i.e.*, unbiasedness) of this procedure is easy to establish, since the expected weighted value of $f$ under $Q$ is just $\mathrm{E}_{Q(x)} f(x) w(x) =$
$$\sum_{x \in X} [f(x)w(x)]\, Q(x) = \sum_{x \in X} \left[ f(x) \frac{P(x)}{Q(x)} \right] Q(x) = \sum_{x \in X} f(x) P(x) = \mathrm{E}_{P(x)} f(x).$$

This technique can be implemented using "indirect" weights $u(x) = \beta P(x)/Q(x)$ and an alternative estimator (Figure 1) that only requires us to compute a fixed multiple of $P(x)$. This preserves asymptotic correctness because $\frac{1}{n} \sum_{i=1}^{n} f(x_i) u(x_i)$ and $\frac{1}{n} \sum_{i=1}^{n} u(x_i)$ converge to $\beta \mathrm{E}_{P(x)} f(x)$ and $\beta$ respectively, which yields $\hat{f} \rightarrow \mathrm{E}_{P(x)} f(x)$ (generally [4]). It will always be possible to apply this extended approach below, but we drop it for now.

Importance sampling is an effective estimation technique when $Q$ approximates $P$ over most of the domain, but it fails when $Q$ misses high probability regions of $P$ and systematically yields samples with small weights. In this case, the resulting estimator will have high variance because the sample will almost always contain unrepresentative points but is sometimes dominated by a few high weight points. To overcome this problem it is critical to obtain data points from the important regions of $P$. Our goal is to avoid generating systematically under-weight samples by *explicitly* searching for significant regions in the target distribution $P$. To do this, and maintain the unbiasedness of the resulting procedure, we develop a series of extensions to importance sampling that are each provably correct.

The first extension is to consider sampling *blocks* of points instead of just individual points. Let $\mathcal{B}$ be a partition of $X$ into finite blocks $B$, where $\bigcup_{B \in \mathcal{B}} B = X$, $B \cap B' = \emptyset$, and each $B$ is finite. (Note that $\mathcal{B}$ can be infinite.) The "block" sampling procedure (Figure 2) draws independent blocks of points to construct the final sample, but then weights points by their target probability $P(x)$ divided by the total block probability $Q(B(x))$. For discrete spaces it is easy to verify that this procedure yields unbiased estimates, since $\mathrm{E}_{Q(x)} \left[ \sum_{x_j \in B(x)} f(x_j) w(x_j) \right] = \sum_{x \in X} \left[ \sum_{x_j \in B(x)} f(x_j) w(x_j) \right] Q(x) =$
$$\sum_{B \in \mathcal{B}} \sum_{x_i \in B} \left[ \sum_{x_j \in B} f(x_j) w(x_j) \right] Q(x_i) = \sum_{B \in \mathcal{B}} \left[ \sum_{x_j \in B} f(x_j) w(x_j) \right] Q(B) =$$
$$\sum_{B \in \mathcal{B}} \left[ \sum_{x_j \in B} f(x_j) \frac{P(x_j)}{Q(B)} \right] Q(B) = \sum_{B \in \mathcal{B}} \left[ \sum_{x_j \in B} f(x_j) P(x_j) \right] = \sum_{x \in X} f(x) P(x).$$

**"Block" importance sampling**
- Draw $x_1, ..., x_n$ independently from $Q$.
- For $x_i$, recover block $B_i = \{x_{i,1}, ..., x_{i,b_i}\}$.
- Create a large sample out of the blocks
$x_{1,1}, ..., x_{1,b_1}, x_{2,1}, ..., x_{2,b_2}, ..., x_{n,1}, ..., x_{n,b_n}$.
- Weight each $x_{i,j}$ by $w(x_{i,j}) = \dfrac{P(x_{i,j})}{\sum_{j=1}^{b_i} Q(x_{i,j})}$.
- For a random variable, $f$, estimate $\mathbb{E}_{P(x)} f(x)$
  by   $\hat{f} = \frac{1}{n} \sum_{i=1}^{n} \sum_{j=1}^{b_i} f(x_{i,j}) w(x_{i,j})$.

**"Sliding window" importance sampling**
- Draw $x_1, ..., x_n$ independently from $Q$.
- For $x_i$, recover block $B_i$, and let $x_{i,1} = x_i$:
  - Get $x_{i,1}$'s successors $x_{i,1}, x_{i,2}, ..., x_{i,m}$ by climbing up $m-1$ steps from $x_{i,1}$.
  - Get predecessors $x_{i,-m+1}, , ..., x_{i,-1}, x_{i,0}$ by climbing down $m-1$ steps from $x_{i,1}$.
  - Weight $w(x_{i,j}) = P(x_{i,j}) / \sum_{k=j-m+1}^{j} Q(x_{i,k})$
- Create final sample from *successor* points
$x_{1,1}, ..., x_{1,m}, x_{2,1}, ..., x_{2,m}, ..., x_{n,1}, ..., x_{n,m}$.
- For a random variable, $f$, estimate $\mathbb{E}_{P(x)} f(x)$
  by   $\hat{f} = \frac{1}{n} \sum_{i=1}^{n} \sum_{j=1}^{m} f(x_{i,j}) w(x_{i,j})$.

Figure 2: "Block" and "sliding window" importance sampling procedures

Crucially, this argument does not depend on *how* the partition of $X$ is chosen. In fact, we could fix any partition, even one that depended on the target distribution $P$, and still obtain an unbiased procedure (so long as the partition remains fixed). Intuitively, this works because blocks are drawn independently from $Q$ and the weighting scheme still produces a "fair" representation of $P$. (Note that the results presented in this paper can all be extended to continuous spaces under mild technical restrictions. However, for the purposes of clarity we will restrict the technical presentation in this paper to the discrete case.)

The second extension is to allow countably infinite blocks that each have a discrete total order $\cdots < x_{i-1} < x_i < x_{i+1} < \cdots$ defined on their elements. This order could reflect the relative probability of $x_i$ and $x_j$ under $P$, but for now we just consider it to be an arbitrary discrete order. To cope with blocks of unbounded length, we employ a "sliding window" sampling procedure that selects a contiguous sub-block of size $m$ from within a larger selected block (Figure 2). This procedure builds each independent subsample by choosing a random point $x_1$ from the proposal distribution $Q$, determining its containing block $B(x_1)$, and then climbing up $m-1$ steps to obtain the successors $x_1, x_2, ..., x_m$, and climbing down $m-1$ steps to obtain the predecessors $x_{-m+1}, ..., x_{-1}, x_0$. The successor points (including $x_1$) appear in the final sample, but the predecessors are only used to determine the weights of the sample points. Weights are determined by the target probability $P(x)$ divided by the probability that the point $x$ appears in a random reconstruction under $Q$. This too yields an unbiased estimator

since $\mathbb{E}_{Q(x)} \left[ \sum_{j=1}^{m} f(x_j) w(x_j) \right] = \sum_{x_\ell \in X} \left[ \sum_{j=\ell}^{\ell+m-1} f(x_j) \frac{P(x_j)}{\sum_{k=j-m+1}^{j} Q(x_k)} \right] Q(x_\ell) =$

$\sum_{B \in \mathcal{B}} \sum_{x_\ell \in B} \sum_{j=\ell}^{\ell+m-1} \frac{f(x_j) P(x_j) Q(x_\ell)}{\sum_{k=j-m+1}^{j} Q(x_k)} = \sum_{B \in \mathcal{B}} \sum_{x_j \in B} \sum_{\ell=j-m+1}^{j} \frac{f(x_j) P(x_j) Q(x_\ell)}{\sum_{k=j-m+1}^{j} Q(x_k)} =$

$\sum_{B \in \mathcal{B}} \sum_{x_j \in B} f(x_j) P(x_j) \frac{\sum_{\ell=j-m+1}^{j} Q(x_\ell)}{\sum_{k=j-m+1}^{j} Q(x_k)} = \sum_{B \in \mathcal{B}} \sum_{x_j \in B} f(x_j) P(x_j) = \sum_{x \in X} f(x) P(x).$

(The middle line breaks the sum into disjoint blocks and then reorders the sum so that instead of first choosing the start point $x_\ell$ and then $x_\ell$'s successors $x_\ell, ..., x_{\ell+m-1}$, we first choose the successor point $x_j$ and then the start points $x_{j-m+1}, ..., x_j$ that could have led to $x_j$). Note that this derivation does not depend on the particular block partition nor on the particular discrete orderings, so long as they remain fixed. This means that, again, we can use partitions and orderings that explicitly depend on $P$ and still obtain a correct procedure.

**"Greedy" importance sampling (1-D)**
- Draw $x_1, ..., x_n$ independently from $Q$.
- For each $x_i$, let $x_{i,1} = x_i$:
  - Compute successors $x_{i,1}, x_{i,2}, ..., x_{i,m}$ by taking $m - 1$ size $\epsilon$ steps in the direction of increase.
  - Compute predecessors $x_{i,-m+1}, ..., x_{i,-1}, x_{i,0}$ by taking $m - 1$ size $\epsilon$ steps in the direction of decrease.
  - If an improper ascent or descent occurs, truncate paths as shown on the upper right.
  - Weight $w(x_{i,j}) = P(x_{i,j}) / \sum_{k=j-m+1}^{j} Q(x_{i,k})$.
- Create the final sample from successor points $x_{1,1}, ..., x_{1,m}, x_{2,1}, ..., x_{2,m}, ..., x_{n,1}, ..., x_{n,m}$.
- For a random variable, $f$, estimate $\mathrm{E}_{P(x)} f(x)$ by $\hat{f} = \frac{1}{n} \sum_{i=1}^{n} \sum_{j=1}^{m} f(x_{i,j}) w(x_{i,j})$.

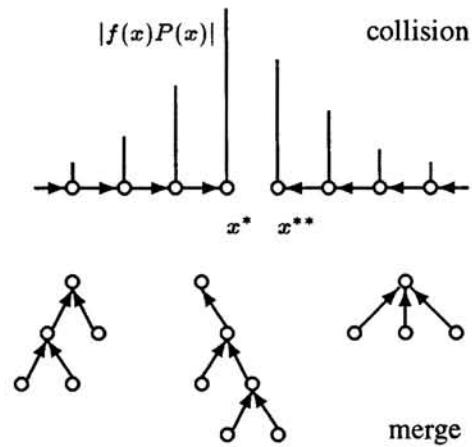

Figure 3: "Greedy" importance sampling procedure; "colliding" and "merging" paths.

## 3   Greedy importance sampling: 1-dimensional case

Finally, we apply the sliding window procedure to conduct an explicit search for important regions in $X$. It is well known that the optimal proposal distribution for importance sampling is just $Q^*(x) = |f(x)P(x)| / \sum_{x \in X} |f(x)P(x)|$ (which minimizes variance [2]). Here we apply the sliding window procedure using an order structure that is determined by the objective $|f(x)P(x)|$. The hope is to obtain reduced variance by sampling independent blocks of points where each block (by virtue of being constructed via an explicit search) is likely to contain at least one or two high weight points. That is, by capturing a moderate size sample of independent high weight points we intuitively expect to outperform standard methods that are unlikely to observe such points by chance. Our experiments below verify this intuition (Figure 4).

The main technical issue is maintaining unbiasedness, which is easy to establish in the 1-dimensional case. In the simple 1-d setting, the "greedy" importance sampling procedure (Figure 3) first draws an initial point $x_1$ from $Q$ and then follows the direction of increasing $|f(x)P(x)|$, taking fixed size $\epsilon$ steps, until either $m - 1$ steps have been taken or we encounter a critical point. A single "block" in our final sample is comprised of a complete sequence captured in one ascending search. To weight the sample points we account for all possible ways each point could appear in a subsample, which, as before, entails climbing *down* $m-1$ steps in the descent direction (to calculate the denominators). The unbiasedness of the procedure then follows directly from the previous section, since greedy importance sampling is equivalent to sliding window importance sampling in this setting.

The only nontrivial issue is to maintain disjoint search paths. Note that a search path must terminate whenever it steps from a point $x^*$ to a point $x^{**}$ with lower value; this indicates that a collision has occurred because some other path must reach $x^*$ from the "other side" of the critical point (Figure 3). At a collision, the largest ascent point $x^*$ must be allocated to a single path. A reasonable policy is to allocate $x^*$ to the path that has the lowest weight penultimate point (but the only critical issue is ensuring that it gets assigned to a single block). By ensuring that the critical point is included in only one of the two distinct search paths, a practical estimator can be obtained that exhibits no bias (Figure 4).

To test the effectiveness of the greedy approach I conducted several 1-dimensional experiments which varied the relationship between $P$, $Q$ and the random variable $f$ (Figure 4). In

these experiments greedy importance sampling strongly outperformed standard methods, including regular importance sampling and directly sampling from the target distribution $P$ (rejection sampling and Metropolis sampling were not competitive). The results not only verify the unbiasedness of the greedy procedure, but also show that it obtains significantly smaller variances across a wide range of conditions. Note that the greedy procedure actually uses $m$ out of $2m - 1$ points sampled for each block and therefore effectively uses a double sample. However, Figure 4 shows that the greedy approach often obtains variance reductions that are far greater than 2 (which corresponds to a standard deviation reduction of $\sqrt{2}$).

## 4   Multi-dimensional case

Of course, this technique is worthwhile only if it can be applied to multi-dimensional problems. In principle, it is straightforward to apply the greedy procedure of Section 3 to multi-dimensional sample spaces. The only new issue is that discrete search paths can now possibly "merge" as well as "collide"; see Figure 3. (Recall that paths could not merge in the previous case.) Therefore, instead of decomposing the domain into a collection of disjoint search paths, the objective $|f(x)P(x)|$ now decomposes the domain into a forest of disjoint search *trees*. However, the same principle could be used to devise an unbiased estimator in this case: one could assign a weight to a sample point $x$ that is just its target probability $P(x)$ divided by the total $Q$-probability of the subtree of points that lead to $x$ in fewer than $m$ steps. This weighting scheme can be shown to yield an unbiased estimator as before. However, the resulting procedure is impractical because in an $N$-dimensional sample space a search tree will typically have a branching factor of $\Omega(N)$; yielding exponentially large trees. Avoiding the need to exhaustively examine such trees is the critical issue in applying the greedy approach to multi-dimensional spaces.

The simplest conceivable strategy is just to ignore merge events. Surprisingly, this turns out to work reasonably well in many circumstances. Note that merges will be a measure zero event in many continuous domains. In such cases one could hope to ignore merges and trust that the probability of "double counting" such points would remain near zero. I conducted simple experiments with a version of greedy importance sampling procedure that ignored merges. This procedure searched in the gradient ascent direction of the objective $|f(x)p(x)|$ and heuristically inverted search steps by climbing in the gradient *descent* direction. Figures 5 and 6 show that, despite the heuristic nature of this procedure, it nevertheless demonstrates credible performance on simple tasks.

The first experiment is a simple demonstration from [12, 10] where the task is to sample from a bivariate Gaussian distribution $P$ of two highly correlated random variables using a "weak" proposal distribution $Q$ that is standard normal (depicted by the elliptical and circular one standard deviation contours in Figure 5 respectively). Greedy importance sampling once again performs very well (Figure 5); achieving unbiased estimates with lower variance than standard Monte Carlo estimators, including common MCMC methods.

To conduct a more significant study, I applied the heuristic greedy method to an inference problem in graphical models: recovering the hidden state sequence from a dynamic probabilistic model, given a sequence of observations. Here I considered a simple Kalman filter model which had one state variable and one observation variable per time-step, and used the conditional distributions $X_t|X_{t-1} \sim N(x_{t-1}, \sigma_s^2)$, $Z_t|X_t \sim N(x_t, \sigma_o^2)$ and initial distribution $X_1 \sim N(0, \sigma_s^2)$. The problem was to infer the value of the final state variable $x_t$ given the observations $z_1, z_2, ..., z_t$. Figure 6 again demonstrates that the greedy approach

has a strong advantage over standard importance sampling. (In fact, the greedy approach can be applied to "condensation" [6, 8] to obtain further improvements on this task, but space bounds preclude a detailed discussion.)

Overall, these preliminary results show that despite the heuristic choices made in this section, the greedy strategy still performs well relative to common Monte Carlo estimators, both in terms of bias and variance (at least on some low and moderate dimension problems). However, the heuristic nature of this procedure makes it extremely unsatisfying. In fact, merge points can easily make up a significant fraction of *finite* domains. It turns out that a rigorously unbiased and feasible procedure can be obtained as follows. First, take greedy fixed size steps in axis parallel directions (which ensures the steps can be inverted). Then, rather than exhaustively explore an entire predecessor tree to calculate the weights of a sample point, use the well known technique of Knuth [9] to sample a *single* path from the root and obtain an unbiased estimate of the total $Q$-probability of the tree. This procedure allows one to formulate an asymptotically unbiased estimator that is nevertheless feasible to implement. It remains important future work to investigate this approach and compare it to other Monte Carlo estimation methods on large dimensional problems—in particular hybrid Monte Carlo [11, 12]. The current results already suggest that the method could have benefits.

# References

[1] P. Dagum and M. Luby. Approximating probabilistic inference in Bayesian belief networks is NP-hard. *Artif Intell*, 60:141–153, 1993.

[2] M. Evans. Chaining via annealing. *Ann Statist*, 19:382–393, 1991.

[3] B. Frey. *Graphical Models for Machine Learning and Digital Communication*. MIT Press, Cambridge, MA, 1998.

[4] J. Geweke. Baysian inference in econometric models using Monte Carlo integration. *Econometrica*, 57:1317–1339, 1989.

[5] W. Gilks, S. Richardson, and D. Spiegelhalter. *Markov chain Monte Carlo in practice*. Chapman and Hall, 1996.

[6] M. Isard and A. Blake. Coutour tracking by stochastic propagation of conditional density. In *ECCV*, 1996.

[7] M. Jordan, Z. Ghahramani, T. Jaakkola, and L. Saul. An introduction to variational methods for graphical models. In *Learning in Graphical Models*. Kluwer, 1998.

[8] K. Kanazawa, D. Koller, and S. Russell. Stochastic simulation algorithms for dynamic probabilistic networks. In *UAI*, 1995.

[9] D. Knuth. Estimating the efficiency of backtracking algorithms. *Math. Comput.*, 29(129):121–136, 1975.

[10] D. MacKay. Intro to Monte Carlo methods. In *Learning in Graphical Models*. Kluwer, 1998.

[11] R. Neal. Probabilistic inference using Markov chain Monte Carlo methods. 1993.

[12] R. Neal. *Bayesian Learning for Neural Networks*. Springer, New York, 1996.

[13] J. Pearl. *Probabilistic Reasoning in Intelligence Systems*. Morgan Kaufmann, 1988.

[14] R. Shacter and M. Peot. Simulation approaches to general probabilistic inference in belief networks. In *Uncertainty in Artificial Intelligence 5*. Elsevier, 1990.

[15] M. Tanner. *Tools for statistical inference: Methods for exploration of posterior distributions and likelihood functions*. Springer, New York, 1993.

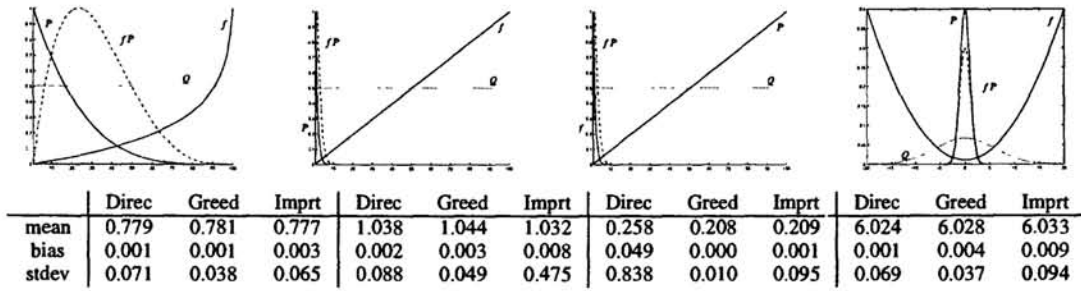

|      | Direc | Greed | Imprt | Direc | Greed | Imprt | Direc | Greed | Imprt | Direc | Greed | Imprt |
|------|-------|-------|-------|-------|-------|-------|-------|-------|-------|-------|-------|-------|
| mean | 0.779 | 0.781 | 0.777 | 1.038 | 1.044 | 1.032 | 0.258 | 0.208 | 0.209 | 6.024 | 6.028 | 6.033 |
| bias | 0.001 | 0.001 | 0.003 | 0.002 | 0.003 | 0.008 | 0.049 | 0.000 | 0.001 | 0.001 | 0.004 | 0.009 |
| stdev| 0.071 | 0.038 | 0.065 | 0.088 | 0.049 | 0.475 | 0.838 | 0.010 | 0.095 | 0.069 | 0.037 | 0.094 |

Figure 4: 1-dimensional experiments: 1000 repetitions on estimation samples of size 100. Problems with varying relationships between $P$, $Q$, $f$ and $|fP|$.

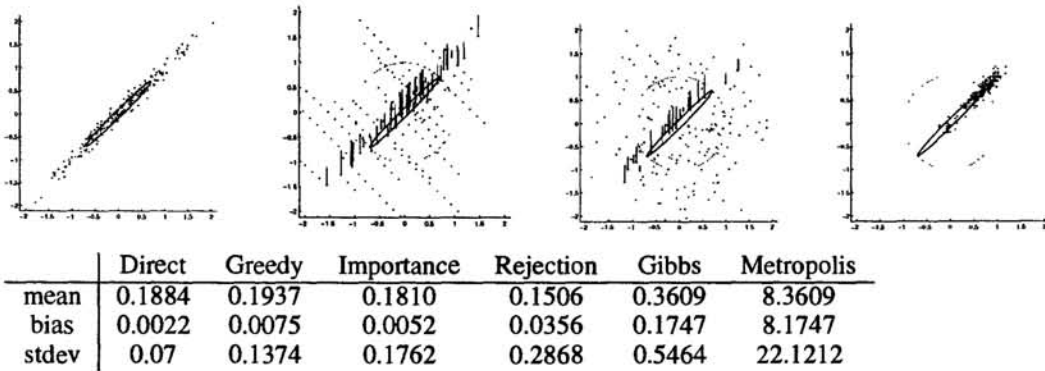

|      | Direct | Greedy | Importance | Rejection | Gibbs | Metropolis |
|------|--------|--------|------------|-----------|-------|------------|
| mean | 0.1884 | 0.1937 | 0.1810 | 0.1506 | 0.3609 | 8.3609 |
| bias | 0.0022 | 0.0075 | 0.0052 | 0.0356 | 0.1747 | 8.1747 |
| stdev| 0.07 | 0.1374 | 0.1762 | 0.2868 | 0.5464 | 22.1212 |

Figure 5: 2-dimensional experiments: 500 repetitions on estimation samples of size 200. Pictures depict: direct, greedy importance, regular importance, and Gibbs sampling, showing 1 standard deviation countours (dots are sample points, vertical lines are weights).

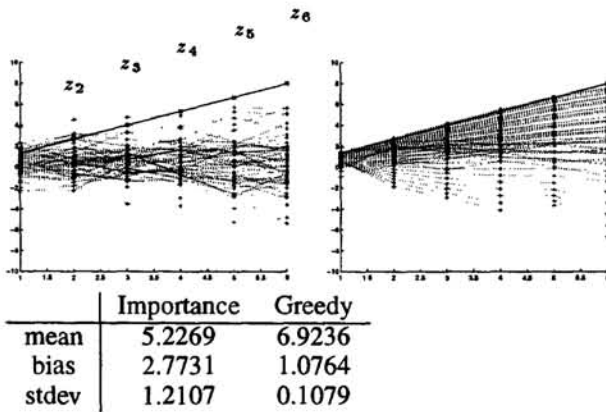

|      | Importance | Greedy |
|------|------------|--------|
| mean | 5.2269 | 6.9236 |
| bias | 2.7731 | 1.0764 |
| stdev| 1.2107 | 0.1079 |

Figure 6: A 6-dimensional experiment: 500 repetitions on estimation samples of size 200. Estimating the value of $x_t$ given the observations $z_1, ..., z_t$. Pictures depict paths sampled by regular versus greedy importance sampling.